# Topmoumoute online natural gradient algorithm

**Nicolas Le Roux**
University of Montreal
nicolas.le.roux@umontreal.ca

**Pierre-Antoine Manzagol**
University of Montreal
manzagop@iro.umontreal.ca

**Yoshua Bengio**
University of Montreal
yoshua.bengio@umontreal.ca

## Abstract

Guided by the goal of obtaining an optimization algorithm that is both fast and yields good generalization, we study the descent direction maximizing the decrease in generalization error or the probability of not increasing generalization error. The surprising result is that from both the Bayesian and frequentist perspectives this can yield the natural gradient direction. Although that direction can be very expensive to compute we develop an efficient, general, online approximation to the natural gradient descent which is suited to large scale problems. We report experimental results showing much faster convergence in computation time and in number of iterations with TONGA (Topmoumoute Online natural Gradient Algorithm) than with stochastic gradient descent, even on very large datasets.

## Introduction

An efficient optimization algorithm is one that quickly finds a good minimum for a given cost function. An efficient learning algorithm must do the same, with the additional constraint that the function is only known through a proxy. This work aims to improve the ability to generalize through more efficient learning algorithms.

Consider the optimization of a cost on a training set with access to a validation set. As the end objective is a good solution with respect to generalization, one often uses early stopping: optimizing the training error while monitoring the validation error to fight overfitting. This approach makes the underlying assumption that overfitting happens at the later stages. A better perspective is that overfitting happens all through the learning, but starts being detrimental only at the point it overtakes the "true" learning. In terms of gradients, the gradient of the cost on the training set is never collinear with the true gradient, and the dot product between the two actually eventually becomes negative. Early stopping is designed to determine when that happens. One can thus wonder: can one limit overfitting before that point? Would this actually postpone that point?

From this standpoint, we discover new justifications behind the natural gradient [1]. Depending on certain assumptions, it corresponds either to the direction minimizing the probability of increasing generalization error, or to the direction in which the generalization error is expected to decrease the fastest. Unfortunately, natural gradient algorithms suffer from poor scaling properties, both with respect to computation time and memory, when the number of parameters becomes large. To address this issue, we propose a generally applicable online approximation of natural gradient that scales linearly with the number of parameters (and requires computation time comparable to stochastic gradient descent). Experiments show that it can bring significant faster convergence and improved generalization.

# 1 Natural gradient

Let $\widetilde{\mathcal{L}}$ be a cost defined as $\widetilde{\mathcal{L}}(\theta) = \int L(x,\theta)p(x)dx$ where $L$ is a loss function over some parameters $\theta$ and over the random variable $x$ with distribution $p(x)$. The problem of minimizing $\widetilde{\mathcal{L}}$ over $\theta$ is often encountered and can be quite difficult. There exist various techniques to tackle it, their efficiency depending on $L$ and $p$. In the case of non-convex optimization, gradient descent is a successful technique. The approach consists in progressively updating $\theta$ using the gradient $\widetilde{g} = \frac{d\widetilde{\mathcal{L}}}{d\theta}$.

[1] showed that the parameter space is a Riemannian space of metric $\widetilde{C}$ (the covariance of the gradients), and introduced the natural gradient as the direction of steepest descent in this space. The natural gradient direction is therefore given by $\widetilde{C}^{-1}\widetilde{g}$. The Riemannian space is known to correspond to the space of functions represented by the parameters (instead of the space of the parameters themselves).

The natural gradient somewhat resembles the Newton method. [6] showed that, in the case of a mean squared cost function, the Hessian is equal to the sum of the covariance matrix of the gradients and of an additional term that vanishes to 0 as the training error goes down. Indeed, when the data are generated from the model, the Hessian and the covariance matrix are equal. There are two important differences: the covariance matrix $\widetilde{C}$ is positive-definite, which makes the technique more stable, but contains no explicit second order information. The Hessian allows to account for variations in the parameters. The covariance matrix accounts for slight variations in the set of training samples. It also means that, if the gradients highly disagree in one direction, one should not go in that direction, even if the mean suggests otherwise. In that sense, it is a conservative gradient.

# 2 A new justification for natural gradient

Until now, we supposed we had access to the true distribution $p$. However, this is usually not the case and, in general, the distribution $p$ is only known through the samples of the training set. These samples define a cost $\mathcal{L}$ (resp. a gradient $g$) that, although close to the true cost (resp. gradient), is not equal to it. We shall refer to $\mathcal{L}$ as the training error and to $\widetilde{\mathcal{L}}$ as the generalization error. The danger is then to overfit the parameters $\theta$ to the training set, yielding parameters that are not optimal with respect to the generalization error.

A simple way to fight overfitting consists in determining the point when the continuation of the optimization on $\mathcal{L}$ will be detrimental to $\widetilde{\mathcal{L}}$. This can be done by setting aside some samples to form a validation set that will provide an independent estimate of $\widetilde{\mathcal{L}}$. Once the error starts increasing on the validation set, the optimization should be stopped. We propose a different perspective on overfitting. Instead of only monitoring the validation error, we consider using as descent direction an estimate of the direction that maximizes the probability of reducing the generalization error. The goal is to limit overfitting at every stage, with the hope that the optimal point with respect to the validation should have lower generalization error.

Consider a descent direction $v$. We know that if $v^T\widetilde{g}$ is negative then the generalization error drops (for a reasonably small step) when stepping in the direction of $v$. Likewise, if $v^Tg$ is negative then the training error drops. Since the learning objective is to minimize generalization error, we would like $v^T\widetilde{g}$ as small as possible, or at least always negative.

By definition, the gradient on the training set is $g = \frac{1}{n}\sum_{i=1}^{n} g_i$ where $g_i = \frac{\partial L(x_i,\theta)}{\partial \theta}$ and $n$ is the number of training samples. With a rough approximation, one can consider the $g_i$s as draws from the true gradient distribution and assume all the gradients are independent and identically distributed. The central limit theorem then gives

$$g \sim \mathcal{N}\left(\widetilde{g}, \frac{\widetilde{C}}{n}\right) \tag{1}$$

where $\widetilde{C}$ is the true covariance matrix of $\frac{\partial L(x,\theta)}{\partial \theta}$ wrt $p(x)$.

We will now show that, both in the Bayesian setting (with a Gaussian prior) and in the frequentist setting (with some restrictions over the type of gradient considered), the natural gradient is optimal in some sense.

## 2.1 Bayesian setting

In the Bayesian setting, $\widetilde{g}$ is a random variable. We would thus like to define a posterior over $\widetilde{g}$ given the samples $g_i$ in order to have a posterior distribution over $v^T \widetilde{g}$ for any given direction $v$. The prior over $\widetilde{g}$ will be a Gaussian centered in 0 of variance $\sigma^2 I$. Thus, using eq. 1, the posterior over $\widetilde{g}$ given the $g_i$s (assuming the only information over $\widetilde{g}$ given by the $g_i$s is through $g$ and $C$) is

$$\widetilde{g}|g,\widetilde{C} \sim \mathcal{N}\left(\left(I + \frac{\widetilde{C}}{n\sigma^2}\right)^{-1} g, \left(\frac{I}{\sigma^2} + n\widetilde{C}^{-1}\right)^{-1}\right) \tag{2}$$

Denoting $\widetilde{C}_\sigma = I + \frac{\widetilde{C}}{n\sigma^2}$, we therefore have

$$v^T \widetilde{g}|g,\widetilde{C} \sim \mathcal{N}\left(v^T \widetilde{C}_\sigma^{-1} g, \frac{v^T \widetilde{C}_\sigma^{-1}\widetilde{C} v}{n}\right) \tag{3}$$

Using this result, one can choose between several strategies, among which two are of particular interest:

- choosing the direction $v$ such that the expected value of $v^T \widetilde{g}$ is the lowest possible (to maximize the immediate gain). In this setting, the direction $v$ to choose is

$$v \propto -\widetilde{C}_\sigma^{-1} g. \tag{4}$$

If $\sigma < \infty$, this is the regularized natural gradient. In the case of $\sigma = \infty$, $\widetilde{C}_\sigma = I$ and this is the batch gradient descent.

- choosing the direction $v$ to minimize the probability of $v^T \widetilde{g}$ to be positive. This is equivalent to finding

$$\text{argmin}_v \frac{v^T \widetilde{C}_\sigma^{-1} g}{\sqrt{v^T \widetilde{C}_\sigma^{-1}\widetilde{C} v}}$$

(we dropped $n$ for the sake of clarity, since it does not change the result). If we square this quantity and take the derivative with respect to $v$, we find $2\widetilde{C}_\sigma^{-1} g (v^T \widetilde{C}_\sigma^{-1} g)(v^T \widetilde{C}_\sigma^{-1}\widetilde{C} v) - 2\widetilde{C}_\sigma^{-1}\widetilde{C} v(v^T \widetilde{C}_\sigma^{-1} g)^2$ at the numerator. The first term is in the span of $\widetilde{C}_\sigma^{-1} g$ and the second one is in the span of $\widetilde{C}_\sigma^{-1}\widetilde{C} v$. Hence, for the derivative to be zero, we must have $g \propto \widetilde{C} v$ (since $\widetilde{C}$ and $\widetilde{C}_\sigma$ are invertible), i.e.

$$v \propto -\widetilde{C}^{-1} g. \tag{5}$$

This direction is the natural gradient and does not depend on the value of $\sigma$.

## 2.2 Frequentist setting

In the frequentist setting, $\widetilde{g}$ is a fixed unknown quantity. For the sake of simplicity, we will only consider (as all second-order methods do) the directions $v$ of the form $v = M^T g$ (i.e. we are only allowed to go in a direction which is a linear function of $g$).

Since $g \sim \mathcal{N}\left(\widetilde{g}, \frac{\widetilde{C}}{n}\right)$, we have

$$v^T \widetilde{g} = g^T M g \sim \mathcal{N}\left(\widetilde{g}^T M \widetilde{g}, \frac{\widetilde{g}^T M^T \widetilde{C} M \widetilde{g}}{n}\right) \tag{6}$$

The matrix $M^*$ which minimizes the probability of $v^T \widetilde{g}$ to be positive satisfies

$$M^* = \text{argmin}_M \frac{\widetilde{g}^T M \widetilde{g}}{\widetilde{g}^T M^T C M \widetilde{g}} \tag{7}$$

The numerator of the derivative of this quantity is $\widetilde{g}\widetilde{g}^T M^T \widetilde{C} M \widetilde{g}\widetilde{g}^T - 2\widetilde{C} M \widetilde{g}\widetilde{g}^T M \widetilde{g}\widetilde{g}^T$. The first term is in the span of $\widetilde{g}$ and the second one is in the span of $\widetilde{C} M \widetilde{g}$. Thus, for this derivative to be 0 for all $\widetilde{g}$, one must have $M \propto \widetilde{C}^{-1}$ and we obtain the same result as in the Bayesian case: the natural gradient represents the direction minimizing the probability of increasing the generalization error.

# 3 Online natural gradient

The previous sections provided a number of justifications for using the natural gradient. However, the technique has a prohibitive computational cost, rendering it impractical for large scale problems. Indeed, considering $p$ as the number of parameters and $n$ as the number of examples, a direct batch implementation of the natural gradient is $O(p^2)$ in space and $O(np^2 + p^3)$ in time, associated respectively with the gradients' covariance storage, computation and inversion. This section reviews existing low complexity implementations of the natural gradient, before proposing TONGA, a new low complexity, online and generally applicable implementation suited to large scale problems. In the previous sections we assumed the true covariance matrix $\widetilde{C}$ to be known. In a practical algorithm we of course use an empirical estimate, and here this estimate is furthermore based on a low-rank approximation denoted $C$ (actually a sequence of estimates $C_t$).

## 3.1 Low complexity natural gradient implementations

[9] proposes a method specific to the case of multilayer perceptrons. By operating on blocks of the covariance matrix, this approach attains a lower computational complexity[1]. However, the technique is quite involved, specific to multilayer perceptrons and requires two assumptions: Gaussian distributed inputs and a number of hidden units much inferior to that of input units. [2] offers a more general approach based on the Sherman-Morrison formula used in Kalman filters: the technique maintains an empirical estimate of the inversed covariance matrix that can be updated in $O(p^2)$. Yet the memory requirement remains $O(p^2)$. It is however not necessary to compute the inverse of the gradients' covariance, since one only needs its product with the gradient. [10] offers two approaches to exploit this. The first uses conjugate gradient descent to solve $Cv = g$. The second revisits [9] thereby achieving a lower complexity. [8] also proposes an iterative technique based on the minimization of a different cost. This technique is used in the minibatch setting, where $Cv$ can be computed cheaply through two matrix vector products. However, estimating the gradient covariance only from a small number of examples in one minibatch yields unstable estimation.

## 3.2 TONGA

Existing techniques fail to provide an implementation of the natural gradient adequate for the large scale setting. Their main failings are with respect to computational complexity or stability. TONGA was designed to address these issues, which it does this by maintaining a low rank approximation of the covariance and by casting both problems of finding the low rank approximation and of computing the natural gradient in a lower dimensional space, thereby attaining a much lower complexity. What we exploit here is that although a covariance matrix needs many gradients to be estimated, we can take advantage of an observed property that it generally varies smoothly as training proceeds and moves in parameter space.

### 3.2.1 Computing the natural gradient direction between two eigendecompositions

Even though our motivation for the use of natural gradient implied the covariance matrix of the empirical gradients, we will use the second moment (i.e. the uncentered covariance matrix) throughout the paper (and so did Amari in his work). The main reason is numerical stability. Indeed, in the batch setting, we have (assuming $C$ is the centered covariance matrix and $g$ the mean) $v = C^{-1}g$, thus $Cv = g$. But then, $(C + gg^T)v = g + gg^T v = g(1 + g^T v)$ and

$$(C + gg^T)^{-1}g = \frac{v}{1 + g^T v} = \bar{v} \qquad (8)$$

Even though the direction is the same, the scale changes and the norm of the direction is bounded by $\frac{1}{\|g\| \cos(g,v)}$.

Since TONGA operates using a low rank estimate of the gradients' non-centered covariance, we must be able to update cheaply. When presented with a new gradient, we integrate its information using the following update formula[2]:

$$C_t = \gamma \hat{C}_{t-1} + g_t g_t^T \tag{9}$$

where $C_0 = 0$ and $\hat{C}_{t-1}$ is the low rank approximation at time step $t-1$. $C_t$ is now likely of greater rank, and the problem resides in computing its low rank approximation $\hat{C}_t$. Writing $\hat{C}_{t-1} = X_{t-1} X_{t-1}^T$,

$$C_t = X_t X_t^T \text{ with } X_t = [\sqrt{\gamma} X_{t-1} \quad g_t]$$

With such covariance matrices, computing the (regularized) natural direction $v_t$ is equal to

$$v_t = (C_t + \lambda I)^{-1} g_t = (X_t X_t^T + \lambda I)^{-1} g_t \tag{10}$$

$$v_t = (X_t X_t^T + \lambda I)^{-1} X_t y_t \text{ with } y_t = [0, \ldots 0, 1]^T. \tag{11}$$

Using the Woodbury identity with positive definite matrices [7], we have

$$v_t = X_t (X_t^T X_t + \lambda I)^{-1} y_t \tag{12}$$

If $X_t$ is of size $p \times r$ (with $r < p$, thus yielding a covariance matrix of rank $r$), the cost of this computation is $O(pr^2 + r^3)$. However, since the Gram matrix $G_t = X_t^T X_t$ can be rewritten as

$$G_t = \begin{pmatrix} \gamma X_{t-1}^T X_{t-1} & \sqrt{\gamma} X_{t-1}^T g_t \\ \sqrt{\gamma} g_t^T X_{t-1} & g_t^T g_t \end{pmatrix} = \begin{pmatrix} \gamma G_{t-1} & \sqrt{\gamma} X_{t-1}^T g_t \\ \sqrt{\gamma} g_t^T X_{t-1} & g_t^T g_t \end{pmatrix}, \tag{13}$$

the cost of computing $G_t$ using $G_{t-1}$ reduces to $O(pr + r^3)$. This stresses the need to keep $r$ small.

### 3.2.2 Updating the low-rank estimate of $C_t$

To keep a low-rank estimate of $C_t = X_t X_t^T$, we can compute its eigendecomposition and keep only the first $k$ eigenvectors. This can be made at low cost using its relation to that of $G_t$:

$$G_t = V D V^T$$
$$C_t = (X_t V D^{-\frac{1}{2}}) D (X_t V D^{-\frac{1}{2}})^T \tag{14}$$

The cost of such an eigendecomposition is $O(kr^2 + pkr)$ (for the computation of the eigendecomposition of the Gram matrix and the computation of the eigenvectors, respectively). Since the cost of computing the natural direction is $O(pr + r^3)$, it is computationally more efficient to let the rank of $X_t$ grow for several steps (using formula 12 in between) and then compute the eigendecomposition using

$$C_{t+b} = X_{t+b} X_{t+b}^T \text{ with } X_{t+b} = \left[ \gamma U_t, \quad \gamma^{\frac{b-1}{2}} g_{t+1}, \quad \ldots \quad \gamma^{\frac{1}{2}} g_{t+b-1}, \quad \gamma^{\frac{t+b}{2}} g_{t+b} \right]$$

with $U_t$ the unnormalized eigenvectors computed during the previous eigendecomposition.

### 3.2.3 Computational complexity

The computational complexity of TONGA depends on the complexity of updating the low rank approximation and on the complexity of computing the natural gradient. The cost of updating the approximation is in $O(k(k+b)^2 + p(k+b)k)$ (as above, using $r = k+b$). The cost of computing the natural gradient $v_t$ is in $O(p(k+b) + (k+b)^3)$ (again, as above, using $r = k+b$). Assuming $k + b \ll \sqrt{(p)}$ and $k \le b$, TONGA's total computational cost per each natural gradient computation is then $O(pb)$.

Furthermore, by operating on minibatch gradients of size $b'$, we end up with a cost per example of $O(\frac{bp}{b'})$. Choosing $b = b'$, yields $O(p)$ per example, the same as stochastic gradient descent. Empirical comparison of cpu time also shows comparable CPU time per example, but faster convergence. In our experiments, $p$ was in the tens of thousands, $k$ was less than 5 and $b$ was less than 50.

The result is an approximate natural gradient with low complexity, general applicability and flexibility over the tradeoff between computations and the quality of the estimate.

# 4 Block-diagonal online natural gradient for neural networks

One might wonder if there are better approximations of the covariance matrix $C$ than computing its first $k$ eigenvectors. One possibility is a block-diagonal approximation from which to retain only the first $k$ eigenvectors of every block (the value of $k$ can be different for each block). Indeed, [4] showed that the Hessian of a neural network with one hidden layer trained with the cross-entropy cost converges to a block diagonal matrix during optimization. These blocks are composed of the weights linking all the hidden units to one output unit and all the input units to one hidden unit. Given the close relationship between the Hessian and the covariance matrices, we can assume they have a similar shape during the optimization.

Figure 1 shows the correlation between the standard stochastic gradients of the parameters of a $16 - 50 - 26$ neural network. The first blocks represent the weights going from the input units to each hidden unit (thus 50 blocks of size 17, bias included) and the following represent the weights going from the hidden units to each output unit (26 blocks of size 51). One can see that the block-diagonal approximation is reasonable. Thus, instead of selecting only $k$ eigenvectors to represent the full covariance matrix, we can select $k$ eigenvectors for every block, yielding the same total cost. However, the rank of the approximation goes from $k$ to $k \times$ number of blocks. In the matrices shown in figure 1, which are of size 2176, a value of $k = 5$ yields an approximation of rank 380.

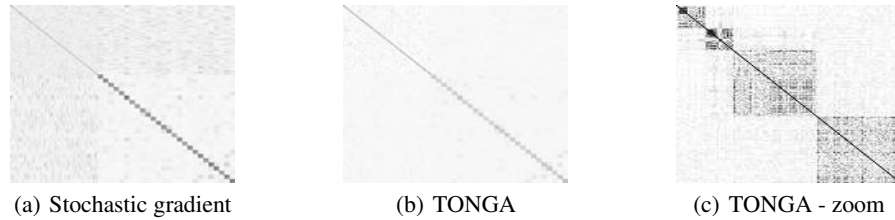

(a) Stochastic gradient      (b) TONGA      (c) TONGA - zoom

Figure 1: Absolute correlation between the standard stochastic gradients after one epoch in a neural network with 16 input units, 50 hidden units and 26 output units when following stochastic gradient directions (left) and natural gradient directions (center and right).

Figure 2 shows the ratio of Frobenius norms $\frac{\|C - \bar{C}\|_F^2}{\|C\|_F^2}$ for different types of approximations $\bar{C}$ (full or block-diagonal). We can first notice that approximating only the blocks yields a ratio of .35 (in comparison, taking only the diagonal of $C$ yields a ratio of .80), even though we considered only 82076 out of the 4734976 elements of the matrix (1.73% of the total). This ratio is almost obtained with $k = 6$. We can also notice that, for $k < 30$, the block-diagonal approximation is much better (in terms of the Frobenius norm) than the full approximation. The block diagonal approximation is therefore very cost effective.

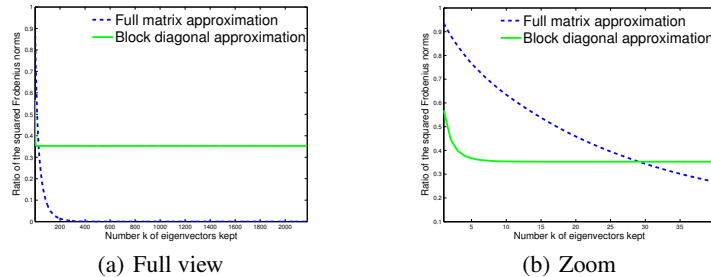

(a) Full view      (b) Zoom

Figure 2: Quality of the approximation $\bar{C}$ of the covariance $C$ depending on the number of eigenvectors kept ($k$), in terms of the ratio of Frobenius norms $\frac{\|C - \bar{C}\|_F^2}{\|C\|_F^2}$, for different types of approximation $\bar{C}$ (full matrix or block diagonal)

This shows the block diagonal approximation constitutes a powerful and cheap approximation of the covariance matrix in the case of neural networks. Yet this approximation also readily applies to any mixture algorithm where we can assume independence between the components.

# 5   Experiments

We performed a small number of experiments with TONGA approximating the full covariance matrix, keeping the overhead of the natural gradient small (ie, limiting the rank of the approximation). Regrettably, TONGA performed only as well as stochastic gradient descent, while being rather sensitive to the hyperparameter values. The following experiments, on the other hand, use TONGA with the block diagonal approximation and yield impressive results. We believe this is a reflection of the phenomenon illustrated in figure 2: the block diagonal approximation makes for a very cost effective approximation of the covariance matrix. All the experiments have been made optimizing hyperparameters on a validation set (not shown here) and selecting the best set of hyperparameters for testing, trying to keep small the overhead due to natural gradient calculations.

One could worry about the number of hyperparameters of TONGA. However, default values of $k = 5$, $b = 50$ and $\gamma = .995$ yielded good results in every experiment. When $\lambda$ goes to infinity, TONGA becomes the standard stochastic gradient algorithm. Therefore, a simple heuristic for $\lambda$ is to progressively tune it down. In our experiments, we only tried powers of ten.

## 5.1   MNIST dataset

The MNIST digits dataset consists of $50000$ training samples, $10000$ validation samples and $10000$ test samples, each one composed of $784$ pixels. There are $10$ different classes (one for every digit).

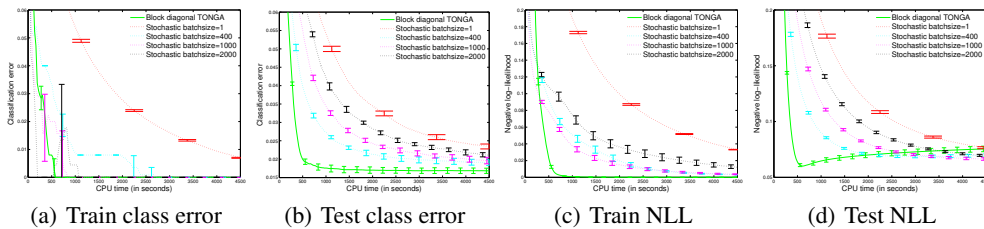

| (a) Train class error | (b) Test class error | (c) Train NLL | (d) Test NLL |

Figure 3: Comparison between stochastic gradient and TONGA on the MNIST dataset (50000 training examples), in terms of training and test classification error and Negative Log-Likelihood (NLL). The mean and standard error have been computed using 9 different initializations.

Figure 3 shows that in terms of training CPU time (which includes the overhead due to TONGA), TONGA allows much faster convergence in training NLL, as well as in testing classification error and testing NLL than ordinary stochastic and minibatch gradient descent on this task. One can also note that minibatch stochastic gradient is able to profit from matrix-matrix multiplications, but this advantage is mainly seen in training classification error.

## 5.2   Rectangles problem

The *Rectangles-images* task has been proposed in [5] to compare deep belief networks and support vector machines. It is a two-class problem and the inputs are $28 \times 28$ grey-level images of rectangles located in varying locations and of different dimensions. The inside of the rectangle and the background are extracted from different real images. We used 900,000 training examples and 10,000 validation examples (no early stopping was performed, we show the whole training/validation curves). All the experiments are performed with a multi-layer network with a 784-200-200-100-2 architecture (previously found to work well on this dataset). Figure 4 shows that in terms of training CPU time, TONGA allows much faster convergence than ordinary stochastic gradient descent on this task, as well as lower classification error.

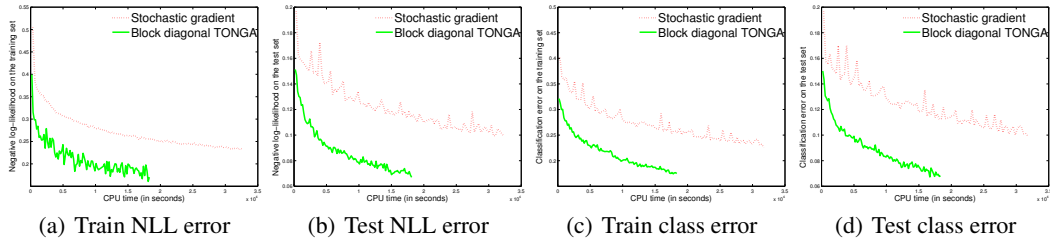

| (a) Train NLL error | (b) Test NLL error | (c) Train class error | (d) Test class error |

Figure 4: Comparison between stochastic gradient descent and TONGA w.r.t. NLL and classification error, on training and validation sets for the rectangles problem (900,000 training examples).

## 6  Discussion

[3] reviews the different gradient descent techniques in the online setting and discusses their respective properties. Particularly, he states that a second order online algorithm (i.e., with a search direction of is $v = Mg$ with $g$ the gradient and $M$ a positive semidefinite matrix) is *optimal* (in terms of convergence speed) when $M$ converges to $H^{-1}$. Furthermore, the speed of convergence depends (amongst other things) on the rank of the matrix $M$. Given the aforementioned relationship between the covariance and the Hessian matrices, the natural gradient is close to optimal in the sense defined above, provided the model has enough capacity. On mixture models where the block-diagonal approximation is appropriate, it allows us to maintain an approximation of much higher rank than a standard low-rank approximation of the full covariance matrix.

## Conclusion and future work

We bring two main contributions in this paper. First, by looking for the descent direction with either the greatest probability of not increasing generalization error or the direction with the largest expected increase in generalization error, we obtain new justifications for the natural gradient descent direction. Second, we present an online low-rank approximation of natural gradient descent with computational complexity and CPU time similar to stochastic gradientr descent. In a number of experimental comparisons we find this optimization technique to beat stochastic gradient in terms of speed and generalization (or in generalization for a given amount of training time). Even though default values for the hyperparameters yield good results, it would be interesting to have an automatic procedure to select the best set of hyperparameters.

## Footnotes

[1]Though the technique allows for a compact representation of the covariance matrix, the working memory requirement remains the same.

[2]The second term is not weighted by $1-\gamma$ so that the influence of $g_t$ in $C_t$ is the same for all $t$, even $t = 0$. To keep the magnitude of the matrix constant, one must use a normalization constant equal to $1 + \gamma + \ldots + \gamma^t$.

## References

[1] S. Amari. Natural gradient works efficiently in learning. *Neural Computation*, 10(2):251–276, 1998.

[2] S. Amari, H. Park, and K. Fukumizu. Adaptive method of realizing natural gradient learning for multilayer perceptrons. *Neural Computation*, 12(6):1399–1409, 2000.

[3] L. Bottou. Stochastic learning. In O. Bousquet and U. von Luxburg, editors, *Advanced Lectures on Machine Learning*, number LNAI 3176 in Lecture Notes in Artificial Intelligence, pages 146–168. Springer Verlag, Berlin, 2004.

[4] R. Collobert. *Large Scale Machine Learning*. PhD thesis, Université de Paris VI, LIP6, 2004.

[5] H. Larochelle, D. Erhan, A. Courville, J. Bergstra, and Y. Bengio. An empirical evaluation of deep architectures on problems with many factors of variation. In *Twenty-fourth International Conference on Machine Learning (ICML'2007)*, 2007.

[6] Y. LeCun, L. Bottou, G. Orr, and K.-R. Müller. Efficient backprop. In G. Orr and K.-R. Müller, editors, *Neural Networks: Tricks of the Trade*, pages 9–50. Springer, 1998.

[7] K. B. Petersen and M. S. Pedersen. The matrix cookbook, feb 2006. Version 20051003.

[8] N. N. Schraudolph. Fast curvature matrix-vector products for second-order gradient descent. *Neural Computation*, 14(7):1723–1738, 2002.

[9] H. H. Yang and S. Amari. Natural gradient descent for training multi-layer perceptrons. Submitted to IEEE Tr. on Neural Networks, 1997.

[10] H. H. Yang and S. Amari. Complexity issues in natural gradient descent method for training multi-layer perceptrons. *Neural Computation*, 10(8):2137–2157, 1998.

